# The Ordered Residual Kernel for Robust Motion Subspace Clustering

**Tat-Jun Chin, Hanzi Wang and David Suter**
School of Computer Science
The University of Adelaide, South Australia
{tjchin, hwang, dsuter}@cs.adelaide.edu.au

## Abstract

We present a novel and highly effective approach for multi-body motion segmentation. Drawing inspiration from robust statistical model fitting, we estimate putative subspace hypotheses from the data. However, instead of ranking them we encapsulate the hypotheses in a novel Mercer kernel which elicits the potential of two point trajectories to have emerged from the same subspace. The kernel permits the application of well-established statistical learning methods for effective outlier rejection, *automatic* recovery of the number of motions and accurate segmentation of the point trajectories. The method operates well under severe outliers arising from spurious trajectories or mistracks. Detailed experiments on a recent benchmark dataset (Hopkins 155) show that our method is superior to other state-of-the-art approaches in terms of recovering the number of motions, segmentation accuracy, robustness against gross outliers and computational efficiency.

## 1   Introduction[1]

Multi-body motion segmentation concerns the separation of motions arising from multiple moving objects in a video sequence. The input data is usually a set of points on the surface of the objects which are tracked throughout the video sequence. Motion segmentation can serve as a useful pre-processing step for many computer vision applications. In recent years the case of rigid (i.e. non-articulated) objects for which the motions could be semi-dependent on each other has received much attention [18, 14, 19, 21, 22, 17]. Under this domain the *affine* projection model is usually adopted. Such a model implies that the point trajectories from a particular motion lie on a linear subspace of at most four, and trajectories from different motions lie on distinct subspaces. Thus multi-body motion segmentation is reduced to the problem of subspace segmentation or clustering.

To realize practical algorithms, motion segmentation approaches should possess four desirable attributes: (1) Accuracy in classifying the point trajectories to the motions they respectively belong to. This is crucial for success in the subsequent vision applications, e.g. object recognition, 3D reconstruction. (2) Robustness against inlier noise (e.g. slight localization error) and gross outliers (e.g. mistracks, spurious trajectories), since getting imperfect data is almost always unavoidable in practical circumstances. (3) Ability to *automatically deduce* the number of motions in the data. This is pivotal to accomplish fully automated vision applications. (4) Computational efficiency. This is integral for the processing of video sequences which are usually large amounts of data.

Recent work on multi-body motion segmentation can roughly be divided into algebraic or factorization methods [3, 19, 20], statistical methods [17, 7, 14, 6, 10] and clustering methods [22, 21, 5]. Notable approaches include Generalized PCA (GPCA) [19, 20], an algebraic method based on the idea that one can fit a union of $m$ subspaces with a set of polynomials of degree $m$. Statistical methods often employ concepts such random hypothesis generation [4, 17], Expectation-Maximization [14, 6]

and geometric model selection [7, 8]. Clustering based methods [22, 21, 5] are also gaining attention due to their effectiveness. They usually include a dimensionality reduction step (e.g. manifold learning [5]) followed by a clustering of the point trajectories (e.g. via spectral clustering in [21]).

A recent benchmark [18] indicated that Local Subspace Affinity (LSA) [21] gave the best performance in terms of *classification accuracy*, although their result was subsequently surpassed by [5, 10]. However, we argue that most of the previous approaches do not simultaneously fulfil the qualities desirable of motion segmentation algorithms. Most notably, although some of the approaches have the means to estimate the number of motions, they are generally unreliable in this respect and require manual input of this parameter. In fact this prior knowledge was given to all the methods compared in [18][2]. Secondly, most of the methods (e.g. [19, 5]) do not explicitly deal with outliers. They will almost always breakdown when given corrupted data. These deficiencies reduce the usefulness of available motion segmentation algorithms in practical circumstances.

In this paper we attempt to bridge the gap between experimental performance and practical usability. Our previous work [2] indicates that robust multi-structure model fitting can be achieved effectively with statistical learning. Here we extend this concept to motion subspace clustering. Drawing inspiration from robust statistical model fitting [4], we estimate random hypotheses of motion subspaces in the data. However, instead of ranking these hypotheses we encapsulate them in a novel Mercer kernel. The kernel can function reliably *despite* overwhelming sampling imbalance, and it permits the application of non-linear dimensionality reduction techniques to effectively identify and reject outlying trajectories. This is then followed by Kernel PCA [11] to maximize the separation between groups and spectral clustering [13] to recover the number of motions and clustering. Experiments on the Hopkins 155 benchmark dataset [18] show that our method is superior to other approaches in terms of the qualities described above, including computational efficiency.

## 1.1 Brief review of affine model multi-body motion segmentation

Let $\{\mathbf{t}_{fp} \in \mathbb{R}^2\}_{p=1,\dots,P}^{f=1,\dots,F}$ be the set of 2D coordinates of $P$ trajectories tracked across $F$ frames. In multi-body motion segmentation the $\mathbf{t}_{fp}$'s correspond to points on the surface of rigid objects which are moving. The goal is to separate the trajectories into groups corresponding to the motion they belong to. In other words, if we arrange the coordinates in the following data matrix

$$\mathbf{T} = \begin{bmatrix} \mathbf{t}_{11} & \cdots & \mathbf{t}_{1P} \\ \vdots & \ddots & \vdots \\ \mathbf{t}_{F1} & \dots & \mathbf{t}_{FP} \end{bmatrix} \in \mathbb{R}^{2F \times P}, \tag{1}$$

the goal is to find the permutation $\mathbf{\Gamma} \in \mathbb{R}^{P \times P}$ such that the columns of $\mathbf{T} \cdot \mathbf{\Gamma}$ are arranged according to the respective motions they belong to. It turns out that under affine projection [1, 16] trajectories from the same motion lie on a distinct subspace in $\mathbb{R}^{2F}$, and each of these *motion subspaces* is of dimensions 2, 3 or 4. Thus motion segmentation can be accomplished via clustering subspaces in $\mathbb{R}^{2F}$. See [1, 16] for more details. Realistically actual motion sequences might contain trajectories which do not correspond to valid objects or motions. These trajectories behave as outliers in the data and, if not taken into account, can be seriously detrimental to subspace clustering algorithms.

## 2 The Ordered Residual Kernel (ORK)

First, we take a statistical model fitting point of view to motion segmentation. Let $\{x_i\}_{i=1,\dots,N}$ be the set of $N$ samples on which we want to perform model fitting. We randomly draw $p$-subsets from the data and use it to fit a hypothesis of the model, where $p$ is the number of parameters that define the model. In motion segmentation, the $x_i$'s are the columns of matrix $\mathbf{T}$, and $p = 4$ since the model is a four-dimensional subspace[3]. Assume that $M$ of such random hypotheses are drawn.

For each data point $x_i$ compute its *absolute* residual set $\mathbf{r}_i = \{r_1^i, \dots, r_M^i\}$ as measured to the $M$ hypotheses. For motion segmentation, the residual is the orthogonal distance to a hypothesis

subspace. We sort the elements in $\mathbf{r}_i$ to obtain the sorted residual set $\tilde{\mathbf{r}}_i = \{r^i_{\lambda^i_1}, \dots, r^i_{\lambda^i_M}\}$, where the permutation $\{\lambda^i_1, \dots, \lambda^i_M\}$ is obtained such that $r^i_{\lambda^i_1} \leq \cdots \leq r^i_{\lambda^i_M}$. Define the following

$$\tilde{\boldsymbol{\theta}}_i := \{\lambda^i_1, \dots, \lambda^i_M\} \tag{2}$$

as the sorted hypothesis set of point $x_i$, i.e. $\tilde{\boldsymbol{\theta}}_i$ depicts the order in which $x_i$ becomes the inlier of the $M$ hypotheses as a *fictitious* inlier threshold is increased from 0 to $\infty$. We define the Ordered Residual Kernel (ORK) between two data points as

$$k_{\tilde{r}}(x_{i_1}, x_{i_2}) := \frac{1}{Z} \sum_{t=1}^{M/h} z_t \cdot k^t_{\cap}(\tilde{\boldsymbol{\theta}}_{i_1}, \tilde{\boldsymbol{\theta}}_{i_2}), \tag{3}$$

where $z_t = \frac{1}{t}$ are the harmonic series and $Z = \sum_{t=1}^{M/h} z_t$ is the $(M/h)$-th harmonic number. Without lost of generality assume that $M$ is wholly divisible by $h$. Step size $h$ is used to obtain the Difference of Intersection Kernel (DOIK)

$$k^t_{\cap}(\tilde{\boldsymbol{\theta}}_{i_1}, \tilde{\boldsymbol{\theta}}_{i_2}) := \frac{1}{h}(|\tilde{\boldsymbol{\theta}}^{1:\alpha_t}_{i_1} \cap \tilde{\boldsymbol{\theta}}^{1:\alpha_t}_{i_2}| - |\tilde{\boldsymbol{\theta}}^{1:\alpha_{t-1}}_{i_1} \cap \tilde{\boldsymbol{\theta}}^{1:\alpha_{t-1}}_{i_2}|) \tag{4}$$

where $\alpha_t = t \cdot h$ and $\alpha_{t-1} = (t-1) \cdot h$. Symbol $\tilde{\boldsymbol{\theta}}^{a:b}_i$ indicates the set formed by the $a$-th to the $b$-th elements of $\tilde{\boldsymbol{\theta}}_i$. Since the contents of the sorted hypotheses set are merely permutations of $\{1 \dots M\}$, i.e. there are no repeating elements,

$$0 \leq k_{\tilde{r}}(x_{i_1}, x_{i_2}) \leq 1. \tag{5}$$

Note that $k_{\tilde{r}}$ is independent of the type of model to be fitted, thus it is applicable to generic statistical model fitting problems. However, we concentrate on motion subspaces in this paper.

Let $\tau$ be a *fictitious* inlier threshold. The kernel $k_{\tilde{r}}$ captures the intuition that, if $\tau$ is low, two points arising from the same subspace will have high normalized intersection since they share many common hypotheses which correspond to that subspace. If $\tau$ is high, implausible hypotheses fitted on outliers start to dominate and decrease the normalized intersection. Step size $h$ allows us to quantify the rate of change of intersection if $\tau$ is increased from 0 to $\infty$, and since $z_t$ is decreasing, $k_{\tilde{r}}$ will evaluate to a high value for two points from the same subspace. In contrast, $k_{\tilde{r}}$ is always low for points not from the same subspace or that are outliers.

**Proof of satisfying Mercer's condition.** Let $D$ be a fixed domain, and $\mathcal{P}(D)$ be the power set of $D$, i.e. the set of all subsets of $D$. Let $S \subseteq \mathcal{P}(D)$, and $p, q \in S$. If $\mu$ is a measure on $D$, then

$$k_{\cap}(p, q) = \mu(p \cap q), \tag{6}$$

called the intersection kernel, is provably a valid Mercer kernel [12]. The DOIK can be rewritten as

$$
\begin{aligned}
k^t_{\cap}(\tilde{\boldsymbol{\theta}}_{i_1}, \tilde{\boldsymbol{\theta}}_{i_2}) &= \frac{1}{h}(|\tilde{\boldsymbol{\theta}}^{(\alpha_{t-1}+1):\alpha_t}_{i_1} \cap \tilde{\boldsymbol{\theta}}^{(\alpha_{t-1}+1):\alpha_t}_{i_2}| \\
&\quad + |\tilde{\boldsymbol{\theta}}^{1:(\alpha_{t-1})}_{i_1} \cap \tilde{\boldsymbol{\theta}}^{(\alpha_{t-1}+1):\alpha_t}_{i_2}| + |\tilde{\boldsymbol{\theta}}^{(\alpha_{t-1}+1):\alpha_t}_{i_1} \cap \tilde{\boldsymbol{\theta}}^{1:(\alpha_{t-1})}_{i_2}|).
\end{aligned} \tag{7}
$$

If we let $D = \{1 \dots M\}$ be the set of all possible hypothesis indices and $\mu$ be uniform on $D$, each term in Eq. (7) is simply an intersection kernel multiplied by $|D|/h$. Since multiplying a kernel with a positive constant and adding two kernels respectively produce valid Mercer kernels [12], the DOIK and ORK are also valid Mercer kernels.●

Parameter $h$ in $k_{\tilde{r}}$ depends on the number of random hypotheses $M$, i.e. step size $h$ can be set as a ratio of $M$. The value of $M$ can be determined based on the size of the $p$-subset and the size of the data $N$ (e.g. [23, 15]), and thus **$h$ is not contingent on knowledge of the true inlier noise scale or threshold**. Moreover, our experiments in Sec. 4 show that segmentation performance is relatively insensitive to the settings of $h$ and $M$.

## 2.1 Performance under sampling imbalance

Methods based on random sampling (e.g. RANSAC [4]) are usually affected by unbalanced datasets. The probability of simultaneously retrieving $p$ inliers from a particular structure is tiny if points

from that structure represent only a small minority in the data. In an unbalanced dataset the "pure" $p$-subsets in the $M$ randomly drawn samples will be dominated by points from the majority structure in the data. This is a pronounced problem in motion sequences, since there is usually a background "object" whose point trajectories form a large majority in the data. In fact, for motion sequences from the Hopkins 155 dataset [18] with typically about 300 points per sequence, $M$ has to be raised to about 20,000 before a pure $p$-subset from the non-background objects is sampled.

However, ORK can function reliably despite serious sampling imbalance. This is because points from the same subspace are roughly equi-distance to the sampled hypotheses in their vicinity, even though these hypotheses might not pass through that subspace. Moreover, since $z_t$ in Eq. (3) is decreasing only residuals/hypotheses in the vicinity of a point are heavily weighted in the intersection. Fig. 1(a) illustrates this condition. Results in Sec. 4 show that ORK excelled even with $M = 1,000$.

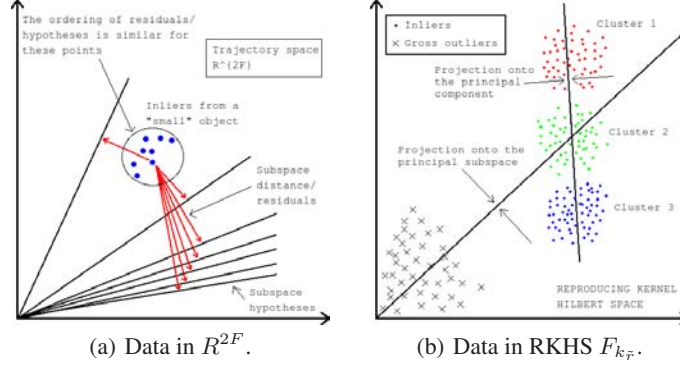

(a) Data in $R^{2F}$.  (b) Data in RKHS $F_{k_{\tilde{r}}}$.

Figure 1: (a) ORK under sampling imbalance. (b) Data in RKHS induced by ORK.

# 3 Multi-Body Motion Segmentation using ORK

In this section, we describe how ORK is used for multi-body motion segmentation.

## 3.1 Outlier rejection via non-linear dimensionality reduction

Denote by $F_{k_{\tilde{r}}}$ the Reproducing Kernel Hilbert Space (RKHS) induced by $k_{\tilde{r}}$. Let matrix $\mathbf{A} = [\phi(x_1) \dots \phi(x_N)]$ contain the input data after it is mapped to $F_{k_{\tilde{r}}}$. The kernel matrix $\mathbf{K} = \mathbf{A}^T \mathbf{A}$ is computed using the kernel function $k_{\tilde{r}}$ as

$$\mathbf{K}_{p,q} = \langle \phi(x_p), \phi(x_q) \rangle = k_{\tilde{r}}(x_p, x_q), \quad p, q \in \{1 \dots N\}. \tag{8}$$

Since $k_{\tilde{r}}$ is a valid Mercer kernel, $\mathbf{K}$ is guaranteed to be positive semi-definite [12]. Let $\mathbf{K} = \mathbf{Q}\boldsymbol{\Delta}\mathbf{Q}^T$ be the eigenvalue decomposition (EVD) of $\mathbf{K}$. Then the rank-$n$ Kernel Singular Value Decomposition (Kernel SVD) [12] of $\mathbf{A}$ is

$$\mathbf{A}^n = [\mathbf{A}\mathbf{Q}^n(\boldsymbol{\Delta}^n)^{-\frac{1}{2}}][(\boldsymbol{\Delta}^n)^{\frac{1}{2}}][(\mathbf{Q}^n)^T] \equiv \mathbf{U}^n\boldsymbol{\Sigma}^n(\mathbf{V}^n)^T. \tag{9}$$

Via the Matlab notation, $\mathbf{Q}^n = \mathbf{Q}_{:,1:n}$ and $\boldsymbol{\Delta}^n = \boldsymbol{\Delta}_{1:n,1:n}$. The left singular vectors $\mathbf{U}^n$ is an orthonormal basis for the $n$-dimensional principal subspace of the *whole dataset* in $F_{k_{\tilde{r}}}$. Projecting the data onto the principal subspace yields

$$\mathbf{B} = [\mathbf{A}\mathbf{Q}^n(\boldsymbol{\Delta}^n)^{-\frac{1}{2}}]^T\mathbf{A} = (\boldsymbol{\Delta}^n)^{\frac{1}{2}}(\mathbf{Q}^n)^T, \tag{10}$$

where $\mathbf{B} = [b_1 \dots b_N] \in \mathbb{R}^{n \times N}$ is the reduced dimension version of $\mathbf{A}$. Directions of the principal subspace are dominated by inlier points, since $k_{\tilde{r}}$ evaluates to a high value generally for them, but always to a low value for gross outliers. Moreover the kernel ensures that points from the same subspace are mapped to the same cluster and vice versa. Fig. 1(b) illustrates this condition.

Fig. 2(a)(left) shows the first frame of sequence "Cars10" from the Hopkins 155 dataset [18] with 100 false trajectories of Brownian motion added to the original data (297 points). The corresponding RKHS norm histogram for $n = 3$ is displayed in Fig. 2(b). The existence of two distinct modes,

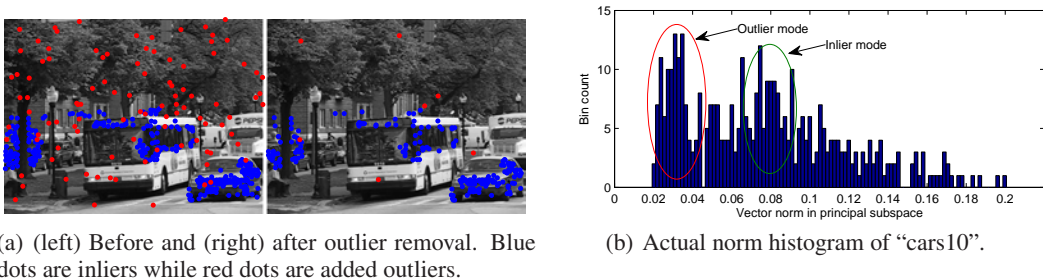

(a) (left) Before and (right) after outlier removal. Blue dots are inliers while red dots are added outliers.

(b) Actual norm histogram of "cars10".

Figure 2: Demonstration of outlier rejection on sequence "cars10" from Hopkins 155.

corresponding respectively to inliers and outliers, is evident. We exploit this observation for outlier rejection by discarding data with low norms in the principal subspace.

The cut-off threshold $\psi$ can be determined by analyzing the shape of the distribution. For instance we can fit a 1D Gaussian Mixture Model (GMM) with two components and set $\psi$ as the point of equal Mahalanobis distance between the two components. However, our experimentation shows that an effective threshold can be obtained by simply setting $\psi$ as the average value of all the norms, i.e.

$$\psi = \frac{1}{N} \sum_{i=1}^{N} \|b_i\|. \tag{11}$$

This method was applied uniformly on all the sequences in our experiments in Sec. 4. Fig. 2(a)(right) shows an actual result of the method on Fig. 2(a)(left).

### 3.2 Recovering the number of motions and subspace clustering

After outlier rejection, we further take advantage of the mapping induced by ORK for recovering the number of motions and subspace clustering. On the remaining data, we perform Kernel PCA [11] to seek the principal components which maximize the variance of the data in the RKHS, as Fig. 1(b) illustrates. Let $\{y_i\}_{i=1,\ldots,N'}$ be the $N'$-point subset of the input data that remains after outlier removal, where $N' < N$. Denote by $\mathbf{C} = [\phi(y_1) \ldots \phi(y_{N'})]$ the data matrix after mapping the data to $F_{k_{\tilde{r}}}$, and by symbol $\tilde{\mathbf{C}}$ the result of adjusting $\mathbf{C}$ with the empirical mean of $\{\phi(y_1), \ldots, \phi(y_{N'})\}$. The *centered* kernel matrix $\tilde{\mathbf{K}}' = \tilde{\mathbf{C}}^T \tilde{\mathbf{C}}$ [11] can be obtained as

$$\tilde{\mathbf{K}}' = \boldsymbol{\nu}^T \mathbf{K}' \boldsymbol{\nu}, \quad \boldsymbol{\nu} = [\mathbf{I}_{N'} - \frac{1}{N'} \mathbf{1}_{N',N'}], \tag{12}$$

where $\mathbf{K}' = \mathbf{C}^T \mathbf{C}$ is the *uncentered* kernel matrix, $\mathbf{I}_s$ and $\mathbf{1}_{s,s}$ are respectively the $s \times s$ identity matrix and a matrix of ones. If $\tilde{\mathbf{K}}' = \mathbf{R} \boldsymbol{\Omega} \mathbf{R}^T$ is the EVD of $\tilde{\mathbf{K}}'$, then we obtain first-$m$ kernel principal components $\mathbf{P}^m$ of $\mathbf{C}$ as the first-$m$ left singular vectors of $\tilde{\mathbf{C}}$, i.e.

$$\mathbf{P}^m = \tilde{\mathbf{C}} \mathbf{R}^m (\boldsymbol{\Omega}^m)^{-\frac{1}{2}}, \tag{13}$$

where $\mathbf{R}^m = \mathbf{R}_{:,1:m}$ and $\boldsymbol{\Omega}_{1:m,1:m}$; see Eq. (9). Projecting the data on the principal components yields

$$\mathbf{D} = [d_1 \ldots d_{N'}] = (\boldsymbol{\Omega}^m)^{\frac{1}{2}} (\mathbf{R}^m)^T, \tag{14}$$

where $\mathbf{D} \in \mathbb{R}^{m \times N'}$. The *affine* subspace $span(\mathbf{P}^m)$ maximizes the spread of the *centered* data in the RKHS, and the projection $\mathbf{D}$ offers an effective representation for clustering. Fig. 3(a) shows the Kernel PCA projection results for $m = 3$ on the sequence in Fig. 2(a).

The number of clusters in $\mathbf{D}$ is recovered via spectral clustering. More specifically we apply the Normalized Cut (Ncut) [13] algorithm. A fully connected graph is first derived from the data, where its weighted adjacency matrix $\mathbf{W} \in \mathbb{R}^{N' \times N'}$ is obtained as

$$\mathbf{W}_{p,q} = \exp(-\|d_p - d_q\|^2 / 2\delta^2), \tag{15}$$

and $\delta$ is taken as the *average nearest neighbour distance* in the Euclidean sense among the vectors in $\mathbf{D}$. The Laplacian matrix [13] is then derived from $\mathbf{W}$ and eigendecomposed. Under Ncut,

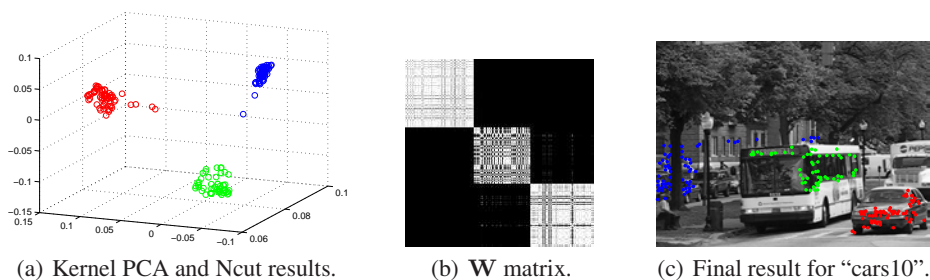

(a) Kernel PCA and Ncut results.    (b) **W** matrix.    (c) Final result for "cars10".

Figure 3: Actual results on the motion sequence in Fig. 2(a)(left).

the number of clusters is revealed as the number of eigenvalues of the Laplacian that are zero or numerically insignificant. With this knowledge, a subsequent $k$-means step is then performed to cluster the points. Fig. 3(b) shows **W** for the input data in Fig. 2(a)(left) after outlier removal. It can be seen that strong affinity exists between points from the same cluster, thus allowing accurate clustering. Figs. 3(a) and 3(c) illustrate the final clustering result for the data in Fig. 2(a)(left).

There are several reasons why spectral clustering under our framework is more successful than previous methods. Firstly, we perform an effective outlier rejection step that removes bad trajectories that can potentially mislead the clustering. Secondly, the mapping induced by ORK deliberately separates the trajectories based on their cluster membership. Finally, we perform Kernel PCA to maximize the variance of the data. Effectively this also improves the separation of clusters, thus facilitating an accurate recovery of the number of clusters and also the subsequent segmentation. This distinguishes our work from previous clustering based methods [21, 5] which tend to operate without maximizing the between-class scatter. Results in Sec. 4 validate our claims.

## 4 Results

Henceforth we indicate the proposed method as "ORK". We leverage on a recently published benchmark on affine model motion segmentation [18] as a basis of comparison. The benchmark was evaluated on the Hopkins 155 dataset[4] which contains 155 sequences with tracked point trajectories. A total of 120 sequences have two motions while 35 have three motions. The sequences contain degenerate and non-degenerate motions, independent and partially dependent motions, articulated motions, nonrigid motions etc. In terms of video content three categories exist: Checkerboard sequences, traffic sequences (moving cars, trucks) and articulated motions (moving faces, people).

### 4.1 Details on benchmarking

Four major algorithms were compared in [18]: Generalized PCA (GPCA) [19], Local Subspace Affinity (LSA) [21], Multi-Stage Learning (MSL) [14] and RANSAC [17]. Here we extend the benchmark with newly reported results from Locally Linear Manifold Clustering (LLMC) [5] and Agglomerative Lossy Compression (ALC) [10, 9]. We also compare our method against Kanatani and Matsunaga's [8] algorithm (henceforth, the "KM" method) in estimating the number of independent motions in the video sequences. Note that KM per se does not perform motion segmentation. For the sake of objective comparisons we use only implementations available publicly[5].

Following [18], motion segmentation performance is evaluated in terms of the labelling error of the point trajectories, where each point in a sequence has a ground truth label, i.e.

$$\text{classification error} = \frac{\text{number of mislabeled points}}{\text{total number of points}}. \quad (16)$$

Unlike [18], we also emphasize on the ability of the methods in recovering *the number of motions*. However, although the methods compared in [18] (except RANSAC) theoretically have the means to

do so, their estimation of the number of motions is generally unrealiable and the benchmark results in [18] were obtained by revealing the actual number of motions to the algorithms. A similar initialization exists in [5, 10] where the results were obtained by giving LLMC and ALC this knowledge a priori (for LLMC, this was given at least to the variant LLMC $4m$ during dimensionality reduction [5], where $m$ is the true number of motions). In the following subsections, where variants exist for the compared algorithms we use results from the best performing variant.

In the following the number of random hypotheses $M$ and step size $h$ for ORK are *fixed* at 1000 and 300 respectively, and unlike the others, ORK is not given knowledge of the number of motions.

## 4.2 Data without gross outliers

We apply ORK on the Hopkins 155 dataset. Since ORK uses random sampling we repeat it 100 times for each sequence and average the results. Table 1 depicts the obtained classification error among those from previously proposed methods. ORK (column 9) gives comparable results to the other methods for sequences with 2 motions (mean = 7.83%, median = 0.41%). For sequences with 3 motions, ORK (mean = 12.62%, median = 4.75%) outperforms GPCA and RANSAC, but is slightly less accurate than the others. *However*, bear in mind that unlike the other methods ORK is not given prior knowledge of the true number of motions and has to estimate this independently.

| Column | 1 | 2 | 3 | 4 | 5 | 6 | 8 | 9 | 10 |
|---|---|---|---|---|---|---|---|---|---|
| Method | REF | GPCA | LSA | MSL | RANSAC | LLMC | ALC | **ORK** | **ORK\*** |
| Sequences with 2 motions | | | | | | | | | |
| Mean | 2.03 | 4.59 | 3.45 | 4.14 | 5.56 | 3.62 | 3.03 | **7.83** | **1.27** |
| Median | 0.00 | 0.38 | 0.59 | 0.00 | 1.18 | 0.00 | 0.00 | **0.41** | **0.00** |
| Sequences with 3 motions | | | | | | | | | |
| Mean | 5.08 | 28.66 | 9.73 | 8.23 | 22.94 | 8.85 | 6.26 | **12.62** | **2.09** |
| Median | 2.40 | 28.26 | 2.33 | 1.76 | 22.03 | 3.19 | 1.02 | **4.75** | **0.05** |

Table 1: Classification error (%) on Hopkins 155 sequences. REF represents the reference/control method which operates based on knowledge of ground truth segmentation. Refer to [18] for details.

We also separately investigate the accuracy of ORK in estimating the number of motions, and compare it against KM [8] which was proposed for this purpose. Note that such an experiment was not attempted in [18] since approaches compared therein generally do not perform reliably in estimating the number of motions. The results in Table 2 (columns 1–2) show that for sequences with two motions, KM (80.83%) outperforms ORK (67.37%) by $\approx 15$ percentage points. However, for sequences with three motions, ORK (49.66%) vastly outperforms KM (14.29%) by more than doubling the percentage points of accuracy. The overall accuracy of KM (65.81%) is slightly better than ORK (63.37%), but this is mostly because sequences with two motions form the majority in the dataset (120 out of 155). This leads us to conclude that ORK is actually the superior method here.

| Dataset | Hopkins 155 | | Hopkins 155 + Outliers | |
|---|---|---|---|---|
| Column | 1 | 2 | 3 | 4 |
| Method | KM | **ORK** | KM | **ORK** |
| 2 motions | 80.83% | **67.37%** | 00.00% | **47.58%** |
| 3 motions | 14.29% | **49.66%** | 100.00% | **50.00%** |
| Overall | 65.81% | **63.37%** | 22.58% | **48.13%** |

Table 2: Accuracy in determining the number of motions in a sequence. Note that in the experiment with outliers (columns 3–4), KM returns a constant number of 3 motions for all sequences.

We re-evaluate the performance of ORK by considering only results on sequences where the number of motions is estimated correctly by ORK (there are about $98 \equiv 63.37\%$ of such cases). The results are tabulated under ORK\* (column 10) in Table 1. It can be seen that when ORK estimates the number of motions correctly, it is significantly more accurate than the other methods.

Finally, we compare the speed of the methods in Table 3. ORK was implemented and run in Matlab on a Dual Core Pentium 3.00GHz machine with 4GB of main memory (this is much less powerful

than the 8 Core Xeon 3.66GHz with 32GB memory used in [18] for the other methods in Table 3). The results show that ORK is comparable to LSA, much faster than MSL and ALC, but slower than GPCA and RANSAC. Timing results of LLMC are not available in the literature.

| Method | GPCA | LSA | MSL | RANSAC | ALC | ORK |
|---|---|---|---|---|---|---|
| 2 motions | 324ms | 7.584s | 11h 4m | 175ms | 10m 32s | **4.249s** |
| 3 motions | 738ms | 15.956s | 1d 23h | 258ms | 10m 32s | **8.479s** |

Table 3: Average computation time on Hopkins 155 sequences.

## 4.3  Data with gross outliers

We next examine the ability of the proposed method in dealing with gross outliers in motion data. For each sequence in Hopkins 155, we add 100 gross outliers by creating trajectories corresponding to mistracks or spuriously occuring points. These are created by randomly initializing 100 locations in the first frame and allowing them to drift throughout the sequence according to Brownian motion. The corrupted sequences are then subjected to the algorithms for motion segmentation. Since only ORK is capable of rejecting outliers, the classification error of Eq. (16) is evaluated on the *inlier* points only. The results in Table 4 illustrate that ORK (column 4) is the most accurate method by a large margin. Despite being given the true number of motions a priori, GPCA, LSA and RANSAC are unable to provide satisfactory segmentation results.

| Column | 1 | 2 | 3 | 4 | 5 |
|---|---|---|---|---|---|
| Method | GPCA | LSA | RANSAC | **ORK** | **ORK\*** |
| Sequences with 2 motions | | | | | |
| Mean | 28.66 | 24.25 | 30.64 | **16.50** | **1.62** |
| Median | 30.96 | 26.51 | 32.36 | **10.54** | **0.00** |
| Sequences with 3 motions | | | | | |
| Mean | 40.61 | 30.94 | 42.24 | **19.99** | **2.68** |
| Median | 41.30 | 27.68 | 43.43 | **8.49** | **0.09** |

Table 4: Classification error (%) on Hopkins 155 sequences with 100 gross outliers per sequence.

In terms of estimating the number of motions, as shown in column 4 in Table 2 the overall accuracy of ORK is reduced to 48.13%. This is contributed mainly by the deterioration in accuracy on sequences with two motions (47.58%), although the accuracy on sequences with three motions are maintained (50.00%). This is not a surprising result, since sequences with three motions generally have more (inlying) point trajectories than sequences with two motions, thus the outlier rates for sequences with three motions are lower (recall that a fixed number of 100 false trajectories are added). On the other hand, the KM method (column 3) is completely overwhelmed by the outliers— for all the sequences with outliers it returned a constant "3" as the number of motions.

We again re-evaluate ORK by considering results from sequences (now with gross outliers) where the number of motions is correctly estimated (there are about $75 \equiv 48.13\%$ of such cases). The results tabulated under ORK\* (column 5) in Table 4 show that the proposed method can accurately segment the point trajectories without being influenced by the gross outliers.

## 5   Conclusions

In this paper we propose a novel and highly effective approach for multi-body motion segmentation. Our idea is based on encapsulating random hypotheses in a novel Mercel kernel and statistical learning. We evaluated our method on the Hopkins 155 dataset with results showing that the idea is superior other state-of-the-art approaches. It is by far the most accurate in terms of estimating the number of motions, and it excels in segmentation accuracy despite lacking prior knowledge of the number of motions. The proposed idea is also highly robust towards outliers in the input data.

**Acknowledgements.** We are grateful to the authors of [18] especially René Vidal for discussions and insights which have been immensely helpful.

## Footnotes

[1]This work was supported by the Australian Research Council (ARC) under the project DP0878801.

[2]As confirmed through private contact with the authors of [18].

[3]Ideally we should also consider degenerate motions with subspace dimensions 2 or 3, but previous work [18] using RANSAC [4] and our results suggest this is not a pressing issue for the Hopkins 155 dataset.

[4]Available at http://www.vision.jhu.edu/data/hopkins155/.

[5]For MSL and KM, see http://www.suri.cs.okayama-u.ac.jp/e-program-separate.html/. For GPCA, LSA and RANSAC, refer to the url for the Hopkins 155 dataset.

# References

[1] T. Boult and L. Brown. Factorization-based segmentation of motions. In *IEEE Workshop on Motion Understanding*, 1991.

[2] T.-J. Chin, H. Wang, and D. Suter. Robust fitting of multiple structures: The statistical learning approach. In *ICCV*, 2009.

[3] J. Costeira and T. Kanade. A multibody factorization method for independently moving objects. *IJCV*, 29(3):159–179, 1998.

[4] M. A. Fischler and R. C. Bolles. Random sample concensus: A paradigm for model fitting with applications to image analysis and automated cartography. *Comm. of the ACM*, 24:381–395, 1981.

[5] A. Goh and R. Vidal. Segmenting motions of different types by unsupervised manifold clustering. In *CVPR*, 2007.

[6] A. Gruber and Y. Weiss. Multibody factorization with uncertainty and missing data using the EM algorithm. In *CVPR*, 2004.

[7] K. Kanatani. Motion segmentation by subspace separation and model selection. In *ICCV*, 2001.

[8] K. Kanatani and C. Matsunaga. Estimating the number of independent motions for multibody segmentation. In *ACCV*, 2002.

[9] Y. Ma, H. Derksen, W. Hong, and J. Wright. Segmentation of multivariate mixed data via lossy coding and compression. *TPAMI*, 29(9):1546–1562, 2007.

[10] S. Rao, R. Tron, Y. Ma, and R. Vidal. Motion segmentation via robust subspace separation in the presence of outlying, incomplete, or corrupted trajectories. In *CVPR*, 2008.

[11] B. Schölkopf, A. Smola, and K. R. Müller. Nonlinear component analysis as a kernel eigenvalue problem. *Neural Computation*, 10:1299–1319, 1998.

[12] J. Shawe-Taylor and N. Cristianini. *Kernel methods for pattern analysis*. Cambridge University Press, 2004.

[13] J. Shi and J. Malik. Normalized cuts and image segmentation. *TPAMI*, 22(8):888–905, 2000.

[14] Y. Sugaya and K. Kanatani. Geometric structure of degeneracy for multi-body motion segmentation. In *Workshop on Statistical Methods in Video Processing*, 2004.

[15] R. Toldo and A. Fusiello. Robust multiple structures estimation with J-Linkage. In *ECCV*, 2008.

[16] C. Tomasi and T. Kanade. Shape and motion from image streams under orthography. *IJCV*, 9(2):137–154, 1992.

[17] P. Torr. Geometric motion segmentation and model selection. *Phil. Trans. Royal Society of London*, 356(1740):1321–1340, 1998.

[18] R. Tron and R. Vidal. A benchmark for the comparison of 3-D motion segmentation algorithms. In *CVPR*, 2007.

[19] R. Vidal and R. Hartley. Motion segmentation with missing data by PowerFactorization and Generalized PCA. In *CVPR*, 2004.

[20] R. Vidal, Y. Ma, and S. Sastry. Generalized Principal Component Analysis (GPCA). *TPAMI*, 27(12):1–15, 2005.

[21] J. Yan and M. Pollefeys. A general framework for motion segmentation: independent, articulated, rigid, non-rigid, degenerate and non-degenerate. In *ECCV*, 2006.

[22] L. Zelnik-Manor and M. Irani. Degeneracies, dependencies and their implications on multibody and multi-sequence factorization. In *CVPR*, 2003.

[23] W. Zhang and J. Kosecká. Nonparametric estimation of multiple structures with outliers. In *Dynamical Vision, ICCV 2005 and ECCV 2006 Workshops*, 2006.

